# Analysis of Short Term Memories for Neural Networks

**Jose C. Principe, Hui-H. Hsu and Jyh-Ming Kuo**

Computational NeuroEngineering Laboratory
Department of Electrical Engineering
University of Florida, CSE 447
Gainesville, FL 32611
principe@synapse.ee.ufl.edu

## Abstract

Short term memory is indispensable for the processing of time varying information with artificial neural networks. In this paper a model for linear memories is presented, and ways to include memories in connectionist topologies are discussed. A comparison is drawn among different memory types, with indication of what is the salient characteristic of each memory model.

## 1  INTRODUCTION

An adaptive system that has to interact with the external world is faced with the problem of coping with the time varying nature of real world signals. Time varying signals, natural or man made, carry information in their time structure. The problem is then one of devising methods and topologies (in the case of interest here, neural topologies) that explore information along time.This problem can be appropriately called *temporal pattern recognition*, as opposed to the more traditional case of static pattern recognition. In static pattern recognition an input is represented by a point in a space with dimensionality given by the number of signal features, while in temporal pattern recognition the inputs are sequence of features. These sequence of features can also be thought as a point but in a vector space of increasing dimensionality. Fortunately the recent history of the input signal is the one that bears more information to the decision making, so the effective dimensionality is finite but very large and unspecified a priori. How to find the appropriate window of input data

(memory depth) for a given application is a difficult problem. Likewise, how to combine the information in this time window to better meet the processing goal is also nontrivial. Since we are interested in adaptive systems, the goal is to let the system find these quantities adaptively using the output error information.

These abstract ideas can be framed more quantitatively in a geometric setting (vector space). Assume that the input is a vector $[u(1),... u(n),....]$ of growing size. The adaptive processor (a neural network in our case) has a fixed size to represent this information, which we assign to its state vector $[x_1(n),....x_N(n)]$ of size N. The usefulness of $x_k(n)$ depends on how well it spans the growing input space (defined by the vector $\mathbf{u}(n)$), and how well it spans the decision space which is normally associated with the minimization of the mean square error (Figure 1). Therefore, in principle, the procedure can be divided into a representational and a mapping problem.

The most general solution to this problem is to consider a nonlinear projection manifold which can be modified to meet both requirements. In terms of neural topologies, this translates to a full recurrent system, where the weights are adapted such that the error criterion is minimized. Experience has shown that this is a rather difficult proposition. Instead, neural network researchers have worked with a wealth of methods that in some way constrain the neural topology.

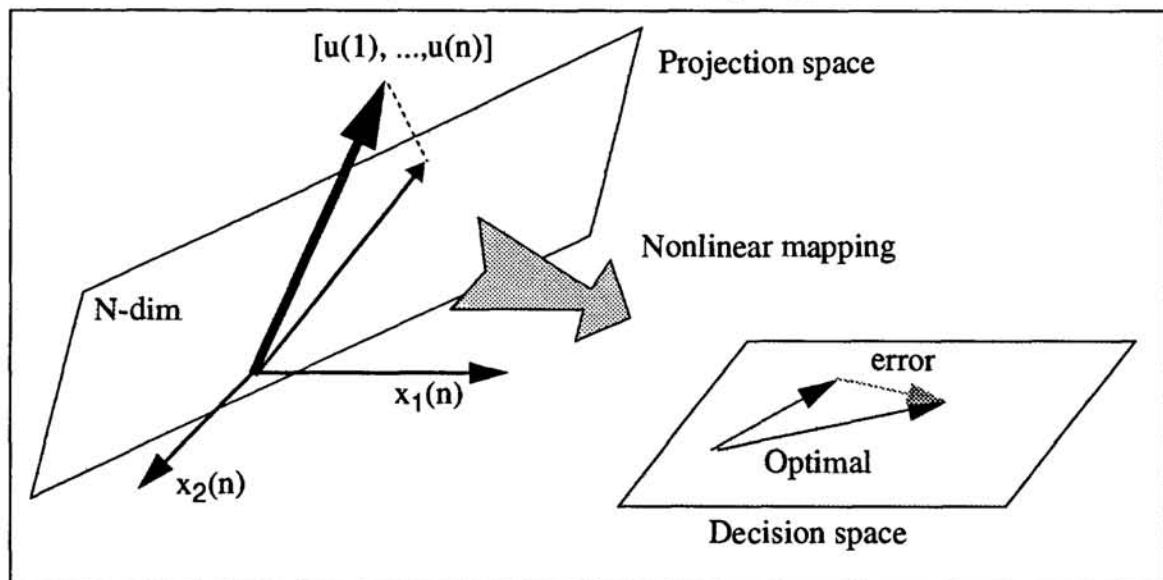

*Figure 1. Projection of u(n) and the error for the task. (for simplicity we are representing only linear manifolds)*

The solution that we have been studying is also constrained. We consider *a linear manifold as the projection space*, which we call the *memory space*. The projection of $\mathbf{u}(n)$ in this space is subsequently mapped by means of a feedforward neural network (multilayer perceptron) to a vector in decision space that minimizes the error criterion. This model gives rise to the focused topologies. The advantage of this constrained model is that it allows an analytical study of the memory structures, since they become linear filters. It is important to stress that the choice of the projection space is crucial for the ultimate performance of the system, because if the projected version of $\mathbf{u}(n)$ in the memory space discards valuable information about $\mathbf{u}(n)$, then

the nonlinear mapping will always produce sub-optimal results.

## 2    Projection in the memory space

If the projection space is linear, then the representational problem can be studied with linear system concepts. The projected vector u(n) becomes $y_n$

$$y_n = \sum_{k=1}^{N} w_k x_{n-k} \tag{1}$$

where $x_n$ are the memory traces. Notice that in this equation the coefficients $w_k$ are independent of time, and their number fixed to N. What is the most general linear structure that implements this projection operation? It is the *generalized feedforward structure* [Principe et al, 1992] (Figure 2), which in connectionist circles has been called the *time lagged recursive network* [Back and Tsoi, 1992]. One can show that the defining relation for generalized feedforward structures is

$$g_k(n) = g(n) \bullet g_{k-1}(n) \qquad k \geq 1$$

where $\bullet$ represents the convolution operation, and $g_0(n) = \delta(n)$. This relation means that the next state vector is constructed from the previous state vector by convolution with the same function g(n), yet unspecified. Different choices of g(n) will provide different choices for the projection space axes. When we apply the input u(n) to this structure, the axes of the projection space become $x_k(n)$, the convolution of u(n) with the tap signals. The projection is obtained by linearly weighting the tap signals according to equation (1).

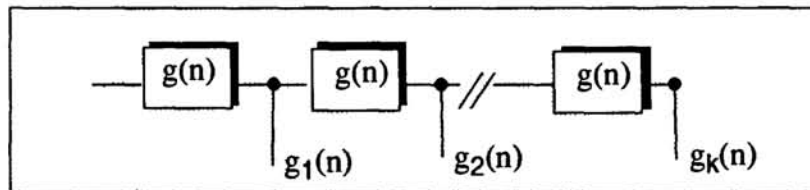

*Figure 2. The generalized feedforward structure*

We define a *memory structure* as a linear system whose generating kernel g(n) is causal $g(n) = 0$ *for* $n < 0$ and normalized, i.e.

$$\sum_{n=0}^{\infty} |g(n)| = 1$$

We define *memory depth* D as the modified center of mass (first moment in time) of the last memory tap.

$$D = \sum_{n=0}^{\infty} n g_k(n)$$

And we define the *memory resolution* R as the number of taps by unit time, which

becomes 1/D. The purpose of the memory structure is to transform the search for an unconstrained number of coefficients (as necessary if we worked directly with u(n)) into one of seeking a fixed number of coefficients in a space with time varying axis.

## 3    Review of connectionist memory structures

The gamma memory [deVries and Principe, 1992] contains as special cases the context unit [Jordan, 1986] and the tap delay line as used in TDNN [Waibel et al, 1989]. However, the gamma memory is also a special case of the generalized feedforward filters where $g(n) = \mu(1-\mu)^n$ which leads to the gamma functions as the tap signals. Figure 3, adapted from [deVries and Principe, 1993], shows the most common connectionist memory structures and its characteristics.

As can be seen when k=1, the gamma memory defaults to the context unit, and when $\mu=1$ the gamma memory becomes the tap delay line. In vector spaces the context unit represents a line, and by changing $\mu$ we are finding the best projection of u(n) on this line. This representation is appropriate when one wants long memories but low resolution.

Likewise, in the tap delay line, we are projecting u(n) in a memory space that is uniquely determined by the input signal, i.e. once the input signal u(n) is set, the axes become u(n-k) and the only degree of freedom is the memory order K. This memory structure has the highest resolution but lacks versatility, since one can only improve the input signal representation by increasing the order of the memory. In this respect, the simple context unit is better (or any memory with a recursive parameter), since the neural system can adapt the parameter $\mu$ to project the input signal for better performance.

We recently proved that the gamma memory structure in continuous time represents a memory space that is rigid [Principe et al, 1994]. When minimizing the output mean square error, the distance between the input signal and the projection space decreases. The recursive parameter in the feedforward structures changes the span of the memory space with respect to the input signal u(n) (which can be visualized as some type of complex rotation). In terms of time domain analysis, the recursive parameter is finding the length of the time window (the memory depth) containing the relevant information to decrease the output mean square error. The recursive parameter $\mu$ can be adapted by gradient descent learning [deVries and Principe, 1992], but the adaptation becomes nonlinear and multiple minima exists.Notice that the memory structure is stable for $0<\mu<2$.

The gamma memory when utilized as a linear adaptive filter extends Widrow's ADALINE [deVries et al, 1992], and results in a more parsimonious filter for echo cancellation [Palkar and Principe, 1994]. Preliminary results with the gamma memory in speech also showed that the performance of word spotters improve when $\mu$ is different from one (i.e. when it is not the tap delay line). In a signal such as speech where time warping is a problem, there is no need to use the full resolution provided by the tap delay line. It is more important to trade depth by resolution.

# 4    Other Memory Structures

There are other memory structures that fit our definition. Back and Tsoi proposed a lattice structure that fits our definition of generalized feedforward structure. Essentially this system orthogonalizes the input, uncorrelating the axis of the vector space (or the signals at the taps of the memory). This method is known to provide the best speed of adaptation because gradient descent becomes Newton's method (after the lattice parameters converge). The problem is that it becomes more computational demanding (more parameters to adapt, and more calculations to perform).

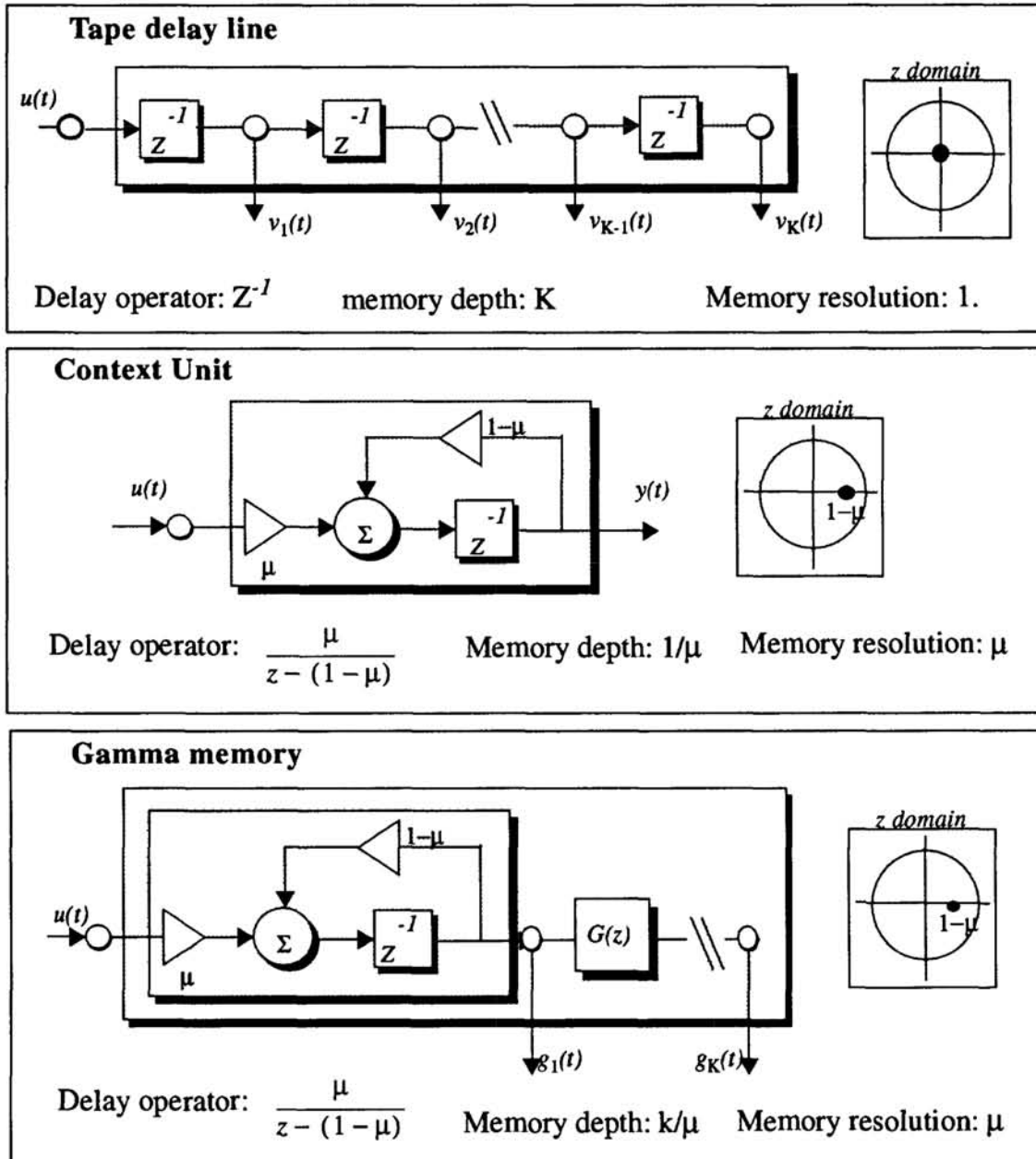

*Figure 3. Connectionist memory structures*

**Laguerre memories**

A set of basis intimately related to the gamma functions is the Laguerre bases. The

Laguerre bases is an orthogonal span of the gamma space [Silva, 1994], which means that the information provided by both memories is the same. The advantage of the Laguerre is that the signals at the taps (the basis) are less correlated and so the adaptation speed becomes faster for values of $\mu$ close to 0 or 2 [Silva, 1994] (the condition number of the matrix created by the tap signals is bounded). Notice that the Laguerre memory is still very easy to compute (a lowpass filter followed by a cascade of first order all pass filters).

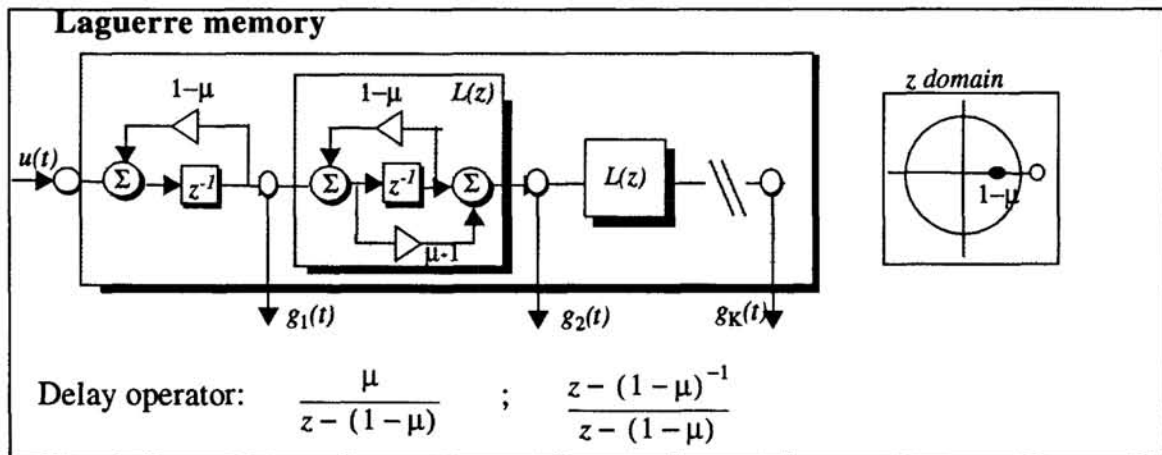

Gamma II memories.

The Gamma memory has a multiple pole that can be adaptively moved along the real Z domain axis, i.e. the Gamma memory can only implement lowpass ($0 < \mu < 1$) or highpass ($1 < \mu < 2$) transfer functions. We experimentally observed that in nonlinear prediction of chaotic time series the recursive parameter sometimes adapts to values less than one. The highpass creates an extra ability to match the prediction by alternating the signs of the samples in the gamma memory (the impulse response for $1 < \mu < 2$ is alternating in sign). But with a single real parameter the adaptation is unable to move the poles to complex values. Two conditions come to mind that require a memory structure with complex poles. First, the information relevant for the signal processing task appears in periodic bursts, and second, the input signal is corrupted by periodic noise. A memory structure with adaptive complex poles can successfully cope with these two conditions by selecting in time the intervals where the information is concentrated (or the windows that do not provide any information for the task). Figure 3 shows one possible implementation for the Gamma II kernel. Notice that for stability, the parameter $\upsilon$ must obey the condition $\mu (1 + \vartheta) < 2$ and $0 < \mu < 2$. Complex poles are obtained for $\upsilon > 0$. These parameters can be adapted by gradient descent [Silva et al, 1992]. In terms of versatility, the Gamma II has a pair of free complex poles, the Gamma I has a pole restricted to the real line in the Z domain, and the tap delay line has the pole set at the origin of the Z domain (z=0). A multilayer perceptron equipped with an input memory layer with the Gamma II memory structure implements a nonlinear mapping on an ARMA model of the input signal.

## 5    How to use Memory structures in Connectionist networks.

Although we have presented this theory with the focused architectures (which

corresponds to a nonlinear moving average model (NMAX)), the memory structures can be placed anywhere in the neural topology. Any nonlinear processing element can feed one of these memory kernels as an extension of [Wan, 1990]. If the memory structures are used to store traces of the output of the net, we obtain a nonlinear autoregressive model (NARX). If they are used both at the input and output, they represent a nonlinear ARMAX model shown very powerful for system identification tasks. When the memory layer is placed in the hidden layers, there is no corresponding linear model.

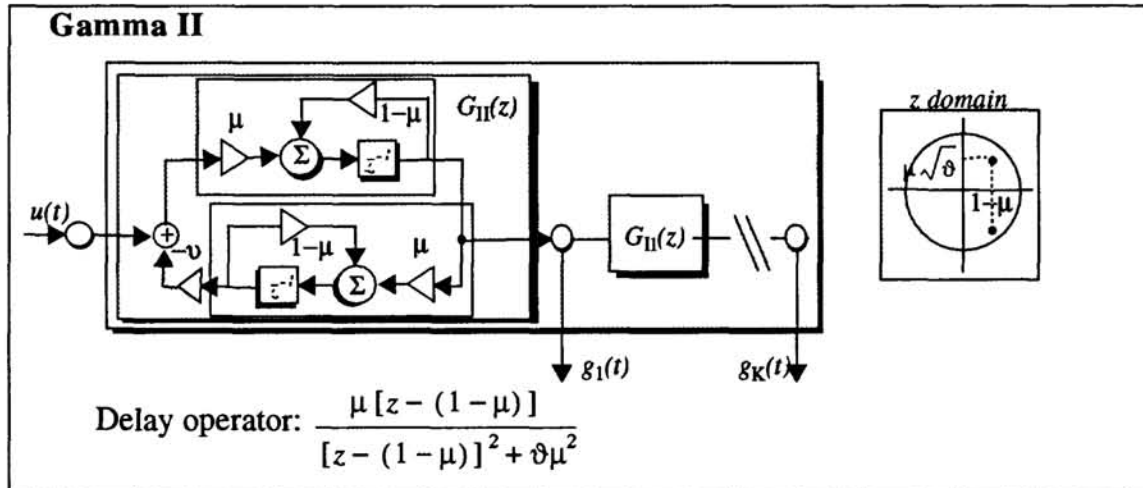

**Gamma II**

Delay operator: $\dfrac{\mu\,[z-(1-\mu)]}{[z-(1-\mu)]^2+\vartheta\mu^2}$

One must realize that these types of memory structures are recursive (except the tap delay line), so their training will involve gradients that depend on time. In the focused topologies the network weights can still be trained with static backpropagation, but the recursive parameter must be trained with real time recurrent learning (RTRL) or backpropagation through time (BPTT). When memory structures are scattered through out the topology, training can be easily accomplished with backpropagation through time, provided a systematic way is utilized to decompose the global dynamics in local dynamics as suggested in [Lefebvre and Principe, 1993].

## 6    Conclusions

The goal of this paper is to present a set of memory structures and show their relationship. The newly introduced Gamma II is the most general of the memories reviewed. By adaptively changing the two parameters $\upsilon,\mu$ the memory can create complex poles at any location in the unit circle. This is probably the most general memory mechanism that needs to be considered. With it one can model poles and zeros of the system that created the signal (if it accepts the linear model).

In this paper we addressed the general problem of extracting patterns in time. We have been studying this problem by pre-wiring the additive neural model, and decomposing it in a linear part -the memory space- that is dedicated to the storage of past values of the input (output or internal states), and in a nonlinear part which is static. The memory space accepts local recursion, which creates a powerful representational structure and where stability can be easily enforced (test in a single parameter). Recursive memories have the tremendous advantage of being able to trade memory depth by resolution. In vector spaces this means changing the relative

position between the projection space and the input signal. However, the problem of finding the best resolution is still open (this means adaptively finding k, the memory order). Likewise ways to adaptively find the optimal value of the memory depth need improvements since the gradient procedures used up to now may be trapped in local minima. It is still necessary to modify the definition of memory depth such that it applies to both of these new memory structures. The method is to define it as the center of mass of the envelope of the last kernel.

**Acknowledgments:** This work was partially supported by NSF grant ECS #920878.

## 7    References

Back, A. D. and A. C. Tsoi, An Adaptive Lattice Architecture for Dynamic Multilayer Perceptrons, Neural Computation, vol. 4, no. 6, pp. 922-931, November, 1992.

de Vries, B. and J. C. Principe, "The gamma model - a new neural model for temporal processing," Neural Networks, vol. 5, no. 4, pp. 565-576, 1992.

de Vries, B., J.C. Principe, and P.G. De Oliveira, "Adaline with adaptive recursive memory," Proc. IEEE Workshop Neural Networks on Signal Processing, Princeton, NJ, 1991.

Jordan, M., "Attractor dynamics and parallelism in a connectionist sequential machine," Proc. 8th annual Conf. on Cognitive Science Society, pp. 531-546, 1986.

Lefebvre, C., and J.C. Principe, "Object-oriented artificial neural network implementations", Proc. World Cong on Neural Nets, vol IV, pp436-439, 1993.

Principe, J. deVries B., Oliveira P., "Generalized feedforward structures: a new class of adaptive fitlers", ICASSP92, vol IV, 244-248, San Francisco.

Principe, J.C., and B. de Vries, "Short term neural memories for time varying signal classification," in Proc. 26th ASILOMAR Conf., pp. 766-770, 1992.

Principe J. C., J.M. Kuo, and S. Celebi," An Analysis of Short Term Memory Structures in Dynamic Neural Networks", accepted in the special issue of recurrent networks of IEEE Trans. on Neural Networks.

Palkar M., and J.C. Principe, "Echo cancellation with the gamma filter," to be presented at ICASSP, 1994.

Silva, T.O., "On the equivalence between gamma and Laguerre filters," to be presented at ICASSP, 1994.

Silva, T.O., J.C. Principe, and B. de Vries, "Generalized feedforward filters with complex poles," Proc. Second IEEE Conf. Neural Networks for Signal Processing, pp.503-510, 1992.

Waiber, A., "Modular Construction of Time-Delay Neural Networks for Speech Recognition," Neural Computation 1, pp39-46, 1989.

Wan, A. E., "Temporal backpropagation: an efficient algorithm for finite impulse response neural networks," Connectionist Models, Proc. of the 1990 Summer School, pp.131-137, 1990.